# Regularized Greedy Importance Sampling

**Finnegan Southey**     **Dale Schuurmans**     **Ali Ghodsi**
School of Computer Science
University of Waterloo
{fdjsouth,dale,aghodsib}@cs.uwaterloo.ca

## Abstract

Greedy importance sampling is an unbiased estimation technique that reduces the variance of standard importance sampling by explicitly searching for modes in the estimation objective. Previous work has demonstrated the feasibility of implementing this method and proved that the technique is unbiased in both discrete and continuous domains. In this paper we present a reformulation of greedy importance sampling that eliminates the free parameters from the original estimator, and introduces a new regularization strategy that further reduces variance without compromising unbiasedness. The resulting estimator is shown to be effective for difficult estimation problems arising in Markov random field inference. In particular, improvements are achieved over standard MCMC estimators when the distribution has multiple peaked modes.

## 1  Introduction

Many inference problems in graphical models can be cast as determining the expected value of a random variable of interest, $f$, given observations drawn according to a target distribution $P$. That is, we are interested in computing $\mathrm{E}_{P(x)} f(x)$. Unfortunately, in natural situations $P$ is usually not in a form that we can sample from efficiently. For example, in standard Bayesian network inference $P(x)$ corresponds to $\mathrm{P}_B(\mathbf{x}|\mathbf{e})$ for a given assignment to evidence variables $\mathbf{e}$ in a given network $B$. It is usually not possible to sample from this distribution directly, nor efficiently evaluate or even approximate $\mathrm{P}_B(\mathbf{x}|\mathbf{e})$ at given points [2]. It is therefore necessary to consider restricted architectures or heuristic and approximate algorithms to perform these tasks [6, 3]. Among the most convenient and successful techniques for performing inference are *stochastic* methods which are guaranteed to converge to a correct solution in the limit of large random samples [7, 14, 4]. These methods can be easily applied to complex inference problems that overwhelm deterministic approaches. The family of stochastic inference methods can be grouped into the *independent* Monte Carlo methods (importance sampling and rejection sampling [7, 4]) and the *dependent* Markov Chain Monte Carlo (MCMC) methods (Gibbs sampling, Metropolis sampling, and Hybrid Monte Carlo) [7, 5, 8, 14]. The goal of all these methods is to simulate drawing a random sample from a target distribution $P(x)$ defined by a graphical model that is hard to sample from directly.

In this paper we improve the *greedy importance sampling* (GIS) technique introduced in [12, 11]. GIS attempts to improve the variance of importance sampling by explicitly searching for important regions in the target distribution $P$. Previous work has shown that search

can be incorporated in an importance sampler while maintaining unbiasedness, leading to improved estimation in simple problems. However, the drawbacks of the previous GIS method are that it has free parameters whose settings affect estimation performance, and its importance weights are directed at achieving unbiasedness without necessarily being directed at reducing variance. In this paper, we introduce a new, parameterless form of greedy importance sampling that performs comparably to the previous method given its best parameter settings. We then introduce a new weight calculation scheme that preserves unbiasedness, but provides further variance reduction by "regularizing" the contributions each search path gives to the estimator. We find that the new procedure significantly improves the original technique and achieves competitive results on difficult estimation problems arising in large discrete domains, such as those posed by Boltzmann machines. Below we first review the generalized importance sampling procedure that forms the core of our estimators before describing the innovations that lead to improved estimators.

## 2   Generalized importance sampling

Importance sampling is a useful technique for estimating $\mathrm{E}_{P(x)} f(x)$ when $P$ cannot be sampled from directly. The basic idea is to draw independent points $x_1, ..., x_n$ according to a simple proposal distribution $Q$ but then weight the points according to $w(x) = P(x)/Q(x)$. Assuming that we can evaluate $P(x)$ the weighted sample can be used to estimate desired expectations (Figure 1).[1] The unbiasedness of this procedure is easy to establish, since for a random variable $f$ the expected *weighted* value of $f$ under $Q$ is $\mathrm{E}_{Q(x)} f(x) w(x) = \sum_{x \in X} f(x) w(x) Q(x) = \sum_{x \in X} f(x) \frac{P(x)}{Q(x)} Q(x) = \sum_{x \in X} f(x) P(x) = \mathrm{E}_{P(x)} f(x)$. (For simplicity we will focus on the discrete case in this paper.) The main difficulty with importance sampling is that even though it is an effective estimation technique when $Q$ approximates $P$ over most of the domain, it performs poorly when $Q$ does not have reasonable mass in high probability regions of $P$. A mismatch of this type results in a high variance estimator since the sample will almost always contains unrepresentative points but will intermittently be dominated by a few high weight points. The idea behind *greedy* importance sampling (GIS) [11, 12] is to avoid generating under-weight samples by explicitly searching for significant regions in the target distribution $P$.

To develop a provably unbiased GIS procedure it is useful to first consider a generalization of standard importance sampling that can be proved to yield unbiased estimates: The *generalized* importance sampling procedure introduced in [12] operates by sampling deterministic blocks of points instead of individual points (Figure 1). Here, to each domain point $x_i$ we associate a fixed block $B_i = \{x_{i,1}, ..., x_{i,m_i}\}$, where $m_i$ is the length of block $B_i$. When $x_i$ is drawn from the proposal distribution $Q$ we recover block $B_i$ and add the block points to the sample.[2] Ensuring unbiasedness then reduces to weighting the sampled points appropriately. To this end, [12] introduces an auxiliary weighting scheme that can be used to obtain unbiased estimates: To each pair of points $x_i, x_j$ (such that $x_j \in B_i$) one associates a weight $\alpha(x_i, x_j)$, where intuitively $\alpha(x_i, x_j)$ is the weight that initiating point $x_i$ assigns to sample point $x_j$ in its block $B_i$. The $\alpha(x_i, x_j)$ values can be arbitrary as long

**"Direct" importance sampling**
- Draw $x_1, ..., x_n$ indep. according to $Q$.
- Weight each point by $w(x_i) = \frac{P(x_i)}{Q(x_i)}$.
- Estimate $\mathrm{E}_{P(x)} f(x)$ by

$$\hat{f} = \frac{1}{n} \sum_{i=1}^{n} f(x_i) w(x_i).$$

**"Indirect" importance sampling**
- Draw $x_1, ..., x_n$ indep. according to $Q$.
- Weight each point by $u(x_i) = \frac{\tilde{P}(x_i)}{Q(x_i)}$
  where $\tilde{P} = PZ$ for some unknown $Z$.
- Estimate $\mathrm{E}_{P(x)} f(x)$ by

$$\hat{f} = \frac{\sum_{i=1}^{n} f(x_i) u(x_i)}{\sum_{i=1}^{n} u(x_i)}.$$

**"Generalized" importance sampling**
- Draw $x_1, ..., x_n$ indep. according to $Q$.
- For each $x_i$, recover its block
  $$B_i = \{x_{i,1}, ..., x_{i,m_i}\}.$$
- Create a large sample out of the blocks
  $$x_{1,1}, ..., x_{1,m_1}, \quad ... \quad x_{n,1}, ..., x_{n,m_n}.$$
- Weight $x_j \in B_i$ by $w_i(x_j) = \frac{P(x_j)}{Q(x_i)}\alpha(x_i, x_j)$
- Estimate $\mathrm{E}_{p(x)} f(x)$ by

$$\hat{f} = \frac{1}{n} \sum_{i=1}^{n} \sum_{k=1}^{m_i} f(x_{i,k}) w_i(x_{i,k})$$

(direct form)

Figure 1: Basic importance sampling procedures

as they satisfy

$$\sum_{x_i \in X} \alpha(x_i, x_j) I(x_i, x_j) = 1 \tag{1}$$

for every $x_j$. (Here $I(x_i, x_j)$ indicates $I(x_i, x_j) = 1$ if $x_j \in B_i$ and $I(x_i, x_j) = 0$ if $x_j \notin B_i$.) That is, for each destination point $x_j$, the total of the incoming $\alpha$-weight has to sum to 1. In fact, it is quite easy to prove that this yields unbiased estimates [12] since the expected *weighted* value of $f$ when sampling initiating $x_i$ under $Q$ is

$$\mathrm{E}_{Q(x_i)} \left[ \sum_{x_j \in B_i} f(x_j) w_i(x_j) \right] = \sum_{x_i \in X} \sum_{x_j \in B_i} f(x_j) \frac{P(x_j)}{Q(x_i)} \alpha(x_i, x_j) Q(x_i)$$

$$= \sum_{x_i \in X} \sum_{x_j \in X} I(x_i, x_j) f(x_j) P(x_j) \alpha(x_i, x_j) = \sum_{x_j \in X} \sum_{x_i \in X} I(x_i, x_j) f(x_j) P(x_j) \alpha(x_i, x_j)$$

$$= \sum_{x_j \in X} f(x_j) P(x_j) \sum_{x_i \in X} I(x_i, x_j) \alpha(x_i, x_j) = \sum_{x_j \in X} f(x_j) P(x_j) = \mathrm{E}_{P(x)} f(x)$$

Crucially, this argument does not depend on *how* the block decomposition is chosen or how the $\alpha$-weights are set, so long as they satisfy (1). That is, one could fix any block decomposition and weighting scheme, even one that depends on the target distribution $P$ and random variable $f$, without affecting the unbiasedness of the procedure. Intuitively, this works because the block structure and weighting scheme are fixed *a priori*, and unbiasedness is achieved by sampling blocks and assigning fair weights to the points. The generality of this outcome allows one to consider using a wide range of alternative importance sampling schemes, while employing appropriate $\alpha$-weights to cancel any bias. In particular, we will determine blocks on-line by following deterministic greedy search paths.

## 3   Parameter-free greedy importance sampling

Our first contribution in this paper is to derive an efficient greedy importance sampling (GIS) procedure that involves no free parameters, unlike the proposal in [12]. One key motivating principle behind GIS is to realize that the optimal proposal distribution for estimating $\mathrm{E}_{P(x)} f(x)$ with standard importance sampling is $Q^*(x) = |f(x)P(x)| / \sum_{x \in X} |f(x)P(x)|$, which minimizes the resulting variance [10]. GIS attempts to overcome a poor proposal distribution by explicitly searching for points that maximally increase the objective $|f(x)P(x)|$ (Figure 2). The primary difficulty in implementing GIS is finding ways to assign the auxiliary weights $\alpha(x_i, x_j)$ so that they satisfy the constraint (1). If this can be achieved, the resulting GIS procedure will be unbiased via the arguments of the previous section. However, the $\alpha$-weights must not only satisfy the constraint (1), they must also be efficiently calculable from a given sample.

**"Greedy" importance sampling**

- Draw $x_1, ..., x_n$ independently from $Q$.
- For each $x_i$, let $x_{i,1} = x_i$ and:
- Compute block $B_i = \{x_{i,1}, x_{i,2}, ..., x_{i,m_i}\}$ by taking local steps in the direction of maximum $|f(x)P(x)|$ until a local max.
- Weight each $x_j \in B_i$ by $w_i(x_j) = \frac{P(x_j)}{Q(x_i)} \alpha(x_i, x_j)$ where $\alpha(x_i, x_j)$ is defined in (2).
- Create the final sample from the blocks
    $x_{1,1}, ..., x_{1,m_1}, \quad ... \quad x_{n,1}, ..., x_{n,m_n}$.
- Estimate $\mathrm{E}_{P(x)} f(x)$ by

$$A = \begin{bmatrix} \alpha_{11} & \alpha_{12} & ... & \alpha_{1n} \\ 0 & \alpha_{22} & ... & \alpha_{2n} \\ \vdots & \vdots & \ddots & \vdots \\ 0 & 0 & ... & \alpha_{nn} \end{bmatrix}$$

$$\phi = \begin{bmatrix} f(1)P(1) \\ f(2)P(2) \\ \vdots \\ f(n)P(n) \end{bmatrix}$$

$$\hat{f} = \frac{1}{n} \sum_{i=1}^{n} \sum_{k=1}^{m_i} f(x_{i,k}) w_i(x_{i,k}).$$

Figure 2: "Greedy" importance sampling procedure (left); Section 4 $A$ matrix (right)

A computationally efficient $\alpha$-weighting scheme can be determined by distributing weight in a search tree in a top down manner: Note that to verify (1) for a domain point $x_j$ we have to consider *every* search path that starts at some other point $x_i$ and passes through $x_j$. If the search is deterministic (which we assume) then the set of search paths entering $x_j$ will form a tree. Let $T_j$ denote the tree of points that lead into $x_j$ and let $\alpha(T_j) = \sum_{x_k \in T_j} \alpha(x_k, x_j)$. In principle, the tree will have unbounded depth since the greedy search procedure does not stop until it has reached a local maximum. Therefore, to ensure $\alpha(T_j) = 1$ we distribute weight down the tree from level 0 (the root, $x_j$) to levels $1, 2, ...$ by a convergent series; where for simplicity we set the total weight allocated at level $k$, $\alpha(T_j^k)$, to be $\alpha(T_j^k) = \frac{1}{(k+1)(k+2)}$. This trivially ensures $\sum_{k=0}^{\infty} \alpha(T_j^k) = 1$.[3] (Finite depth bounds will be handled automatically below.)

Having established the total weight at level $k$, $\alpha(T_j^k)$, we must then determine how much of that weight is allocated to a particular point at that level. Given the entire search tree this would be trivial, but the greedy search paths will typically provide only a single branch of the tree. We accomplish the allocation by recursively dividing the weight equally amongst branches, starting at the root of the tree. Thus, if $b_{i+k}$ is the inward branching factor at the root, we divide $\alpha(T_j^k)$ by $b_{i+k}$ at the first level. Then, following the path to a desired point $x_i$, we successively divide the remaining weight at each point by the observed branching factor $b_{i+k-1}$, $b_{i+k-2}$, etc. until we reach $x_i$. In the case $b_i = 0$, $x_i$ has no descendants and we compensate by adding the mass of the missing subtree to $x_i$'s weight. This scheme is efficient to compute because we require only the branching factors along a given search path to correctly allocate the weight. This yields the following weighting scheme that runs in linear time and exactly satisfies the constraint (1): Given a start point $x_i$ and a search path $x_i, x_{i+1}, ..., x_{i+k} = x_j$ from $x_i$ to $x_j$, we assign a weight $\alpha(x_i, x_j)$ by

$$\alpha(x_i, x_j) = \begin{cases} \frac{1}{b_{i+1} b_{i+2} \cdots b_{i+k} (k+1)(k+2)} & \text{if } b_i > 0 \\ \frac{1}{b_{i+1} b_{i+2} \cdots b_{i+k} (k+1)} & \text{if } b_i = 0 \end{cases} \tag{2}$$

where $b_{i+\ell}$ denotes the inward branching factor of point $x_{i+\ell}$. A simple induction proof can be used to show that $\sum_{x_i} \alpha(x_i, x_j) = 1$. Therefore, the new $\alpha$-weighting scheme provides an efficient unbiased method for implementing GIS that does not use any free parameters.

## 4 Variance reduction

While GIS reduces variance by searching, the $\alpha$-weight correction scheme outlined above is designed only to correct bias and does not specifically address variance issues. However,

there is a lot of leeway in setting the $\alpha$-weights since the normalization constraint (1) is quite weak. In fact, one can exploit this additional flexibility to determine minimum variance unbiased estimators in simple cases. To illustrate, consider a toy domain consisting of points $1, 2, 3, ..., n$, where $0 \leq f(i)P(i) < f(i+1)P(i+1)$. Assume the search is constrained to move between adjacent points so that from every initial point the greedy search will move to the right until it hits point $n$. Any $\alpha$-weighting scheme for this domain can be expressed as a matrix, $A$, shown in Figure 2, where row $i$ corresponds to the search block retrieved by starting at point $i$. Note that the constraint (1) amounts to requiring that the *columns* of $A$ sum to 1. However, it is the *rows* of $A$ that correspond to search blocks sampled during estimation. If we assume a uniform proposal distribution $Q = (\frac{1}{n}, ..., \frac{1}{n})^\top$ then $nA\phi$ gives the column vector of block estimates that correspond to each start point. The variance of the overall estimator then becomes equal to the variance of the column vector $nA\phi$. In particular, if each row produces the same estimate, the estimator will have *zero* variance. We conclude that zero variance is achieved iff $nA\phi$ equals a constant. Thus, the unbiasedness constraints behave orthogonally to the zero variance constraints: unbiasedness imposes a constraint on columns of $A$ whereas zero variance imposes a constraint on rows of $A$. An optimal estimator will satisfy both sets of constraints. Since there are $2n$ constraints in total and $n(n+1)/2$ variables, one can apparently solve for a zero variance unbiased estimator (for $n > 2$). However, it turns out that the constraint matrix does not have full rank, and it is not always possible to achieve zero bias and variance for given $\phi$. Nevertheless, one can obtain an optimal GIS estimator by solving a quadratic program for the $A$ which minimizes variance subject to satisfying the linear unbiasedness constraints.

The point of this simple example is not to propose a technique that explicitly enumerates the domain in order to construct a minimum variance GIS estimator. (Although the above discussion applies to any finite domain—all one needs to do is encode the search topology in the weight matrix $A$.) Rather, the point is to show that a significant amount of flexibility remains in setting the $\alpha$-weights—even after the unbiasedness constraints have been satisfied—and that this additional flexibility can be exploited to reduce variance.

We can now extend these ideas to a more realistic, general situation: To reduce the variance of the GIS estimator developed in Section 3, our idea is to equalize the block totals among different search paths. The main challenge is to adjust $\alpha$-weights in a way that equalizes block totals without introducing bias, and without requiring excessive computational overhead. Here we follow the style of local correction employed in Section 3. First note that when traversing a path from $x_i$ to $x_j$, the blocks sampled by GIS produce estimates of the form $W_i = \sum_{\ell=0}^{k} \frac{f(x_{i+\ell})P(x_{i+\ell})}{Q(x_i)} \alpha(x_i, x_{i+\ell})$. Now consider an intermediate point $x_{i+\ell}$ in the search. This point will have been arrived at via some predecessor $x_{i+\ell-1}$, but we could have arrived at $x_{i+\ell}$ via any one of its possible predecessors $x_p$. We would like to equalize the block totals that would have been obtained by arriving via any one of these predecessor points. The key to maintaining unbiasedness is to ensure that any weight calculation performed at a point in a search tree is consistent, regardless of the path taken to reach that point. Since we cannot anticipate the initial points, it is only convenient to equalize the subtotals from the predecessors $x_p$, through $x_{i+\ell}$, and up to the root $x_j$. Let $U_{i+\ell}$ denote the total sum obtained by points after $x_{i+\ell}$; i.e. from $x_{i+\ell+1}$ to $x_j$. We equalize the different predecessor totals by determining factors $\gamma_p$ which satisfy the constraints

$$f(x_p)P(x_p) + \gamma_p(f(x_{i+\ell})P(x_{i+\ell}) + U_{i+\ell}) = \lambda$$

over the predecessors $x_p$. This scales the parent quantity $f(x_{i+\ell})P(x_{i+\ell}) + U_{i+\ell}$ on each path to compensate for differences between predecessors. The equalization and unbiasedness constraints form a linear system whose solution we rescale to obtain positive $\gamma_p$. The $\gamma_p$ are computed starting at the end of the block and working backwards. The results can be easily incorporated into the GIS procedure by multiplying the original $\alpha$-weights in (2) by the product $\gamma_{i+1}\gamma_{i+2}...\gamma_{i+\ell-1}$. Importantly, at a given search point, any of its predecessors will calculate the same $\gamma$-correction scheme locally, regardless of which predecessor

is actually sampled. This means that the correction scheme is not sample-dependent but fixed ahead of time. It is easy to prove that any fixed $\gamma$-weighting scheme that satisfies $\sum_{p=1}^{b_{i+\ell}} \gamma_p = b_{i+\ell}$, and is applied to an unbiased $\alpha$-weighting, will satisfy (1). The benefit of this scheme is that it reduces variance while preserving unbiasedness.[4]

## 5  Empirical results: Markov random field estimation

To investigate the utility of the GIS estimators we conducted experiments on inference problems in Markov random fields. Markov random fields are an important class of undirected graphical model which include Boltzmann machines as a special case [1]. These models are known to pose intractable inference problems for exact methods. Typically, standard MCMC methods such as Gibbs sampling and Metropolis sampling are applied to such problems, but their success is limited owing to the fact that these estimators tend to get trapped in local modes [7]. Moreover, improved MCMC methods such as Hybrid Monte Carlo [8] cannot be directly applied to these models because they require continuous sample spaces, whereas Boltzmann machines and other random field models define distributions on a discrete domain. Standard importance sampling is also a poor estimation strategy for these models because a simple proposal distribution (like uniform) has almost no chance of sampling in relevant regions of the target distribution [7]. Explicitly searching for modes would seem to provide an effective estimation strategy for these problems.

We consider a generalization of Boltzmann machines that defines a joint distribution over a set of discrete variables $x_1, ..., x_N$, $x_i \in \{-1, +1\}$, according to

$$ P(\mathbf{x}) = \exp\left(-\tfrac{1}{T}E(\mathbf{x})\right)/Z \quad \text{where} \quad E(\mathbf{x}) = \sum_{i,j:j>i} g_{ij}(x_i, x_j) + \sum_i g_i(x_i). $$

Here $T$ is the "temperature" of the model and $E(\mathbf{x})$ defines the "energy" of configuration $\mathbf{x}$; the functions $g_{ij}$ and $g_i$ define the local energy between pairs of variables and individual variables respectively; and $Z$ is a normalization constant. Exact inference in such a model is difficult because the normalization constant $Z$ is typically unknown. Moreover, $Z$ is usually not possible to obtain exactly because it is defined as an exponentially large sum that is not prone simplification.[5] We experimented with two classes of generalized Boltzmann machines: generalized *Ising* models, where the underlying graph is a 2 dimensional grid, and *random* models, where the graph is generated by randomly choosing links between variables. For each model, the $g$ function values were chosen randomly from a standard normal distribution. We considered the objective functions $f(\mathbf{x}) = E(\mathbf{x})$ (expected energy); $f(\mathbf{x}) = \sum_i \mathbf{1}(x_i = 1)$ (expected number of 1's in a configuration); and $f(\mathbf{x}) = \sum_{i,j:j>i} \mathbf{1}(x_i = x_j = 1)$ (expected number of pairwise "and's" in a configuration). The latter two objectives are summaries of the quantities needed to estimate gradients in standard Boltzmann machine learning algorithms [1]. This would seem to be an ideal model on which to test our methods.

We conducted experiments by fixing a model and temperature and ran the estimators for a fixed amount of CPU time. Each estimator was re-run 1000 times to estimate their root mean squared error (RMSE) on small models where exact answers could be calculated, or standard deviation (STD) on large models where no such exact answer is feasible. We compared estimators by controlling their *run time* (given a reasonable C implementation) not just their sample size, because the different estimators use different computational overheads, and run time is the only convenient way to draw a fair comparison. For example, GIS methods require a substantial amount of additional computation to find the greedy search

| E(energy) | Avg SS | RMSE @ T=1.0 | T=0.5 | T=0.25 | T=0.1 | T=0.05 | T=0.025 |
|---|---|---|---|---|---|---|---|
| IS | 5094 | 27.75 | 68.96 | 145.97 | 374.04 | 749.42 | 1503.73 |
| GISold | 1139 | 13.89 | 12.93 | 12.96 | 13.35 | 10.46 | 12.59 |
| GISnew | 1015 | 14.31 | 13.73 | 13.94 | 15.25 | 11.78 | 11.03 |
| GISreg | 1015 | 3.01 | 4.10 | 5.57 | 6.61 | 6.20 | 7.72 |
| Gibbs | 36524 | 0.21 | 0.37 | 4.44 | 21.86 | 53.44 | 108.13 |
| Metro | 35885 | 0.28 | 0.53 | 5.75 | 24.56 | 56.16 | 122.46 |

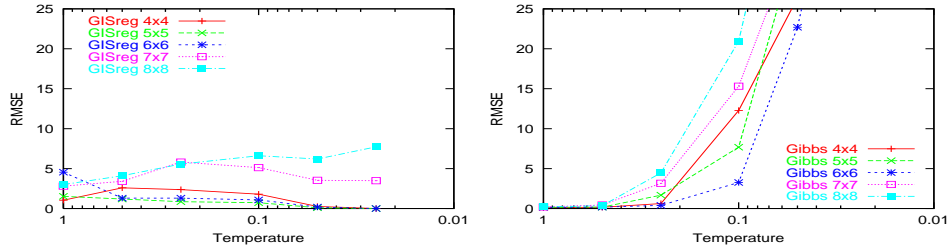

Figure 3: Estimating average energy in a random field model (table shows results for $8 \times 8$).

| E(and's) | Avg SS | RMSE @ T=1.0 | T=0.5 | T=0.25 | T=0.1 | T=0.05 | T=0.025 |
|---|---|---|---|---|---|---|---|
| IS | 4764 | 6.10 | 8.42 | 9.60 | 10.45 | 10.15 | 10.15 |
| GISold | 1125 | 6.33 | 5.16 | 4.03 | 2.57 | 0.64 | 0.43 |
| GISnew | 1015 | 6.09 | 5.16 | 4.30 | 2.85 | 0.61 | 0.15 |
| GISreg | 1015 | 3.56 | 3.06 | 2.43 | 0.90 | 0.17 | 0.05 |
| Gibbs | 22730 | 0.33 | 0.36 | 0.59 | 0.70 | 1.41 | 1.54 |
| Metro | 25789 | 0.37 | 0.43 | 0.63 | 0.76 | 1.30 | 1.41 |

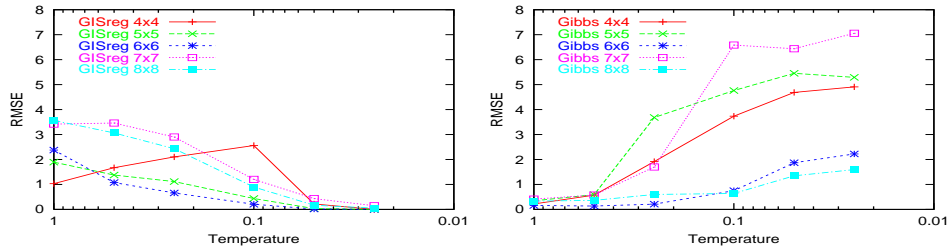

Figure 4: Estimating average "sum of and's" in a random field model (table shows $8 \times 8$).

paths and calculate inward branching factors, and consequently they must use substantially smaller sample sizes than their counterparts to ensure a fair comparison. However, the GIS estimators still seem to obtain reasonable results despite their sample size disadvantage. For the GIS procedures we implemented a simple search that only ascends in $P(\mathbf{x})$ not $|f(\mathbf{x})P(\mathbf{x})|$, and we only used a uniform proposal distribution in all our experiments. We also only report results for the indirect versions of all importance samplers (cf. Figure 1).

Figures 3 and 4 show typical outcomes of our experiments. Table 3 shows results for estimating expected energy in an $8 \times 8$ generalized Ising model when temperature is dropped from 1.0 to 0.025. Figure 4 shows comparable results for estimating the "sum of and's". Standard importance sampling (IS) is a poor estimator in this domain, even when it is able to use 4.5 times as many data points as the GIS estimators. IS becomes particularly poor when the temperature drops. Among GIS estimators, the new, parameter-free version introduced in Section 3 (GIS_new) compares favorably to the previous technique of [12] (GIS_old). The regularized GIS from Section 4 (GIS_reg) is clearly superior to either.

Next, to compare the importance sampling approaches to the MCMC methods, we see the dramatic effect of temperature reduction. Owing to their simplicity (and an efficient implementation), the MCMC samplers were able to gather about 20 to 30 times as many data

points as the GIS estimators in the same amount of time. The effect of this substantial sample size advantage is that the MCMC methods demonstrate far better performance at high temperatures; apparently owing to an evidential advantage. However, as the temperature is lowered, a well known effect takes hold as the the low energy configurations begin to dominate the distribution. At low temperatures the modes around the low energy configurations become increasingly peaked and standard MCMC estimators become trapped in modes from which they are unable to escape [8, 7]. This results in a very poor estimate that is dominated by arbitrary modes. Figures 3 and 4 show the RMSE curves of Gibbs sampling and GIS_reg, side by side, as temperature is decreased in different models. By contrast to MCMC procedures, the GIS procedures exhibit almost no accuracy loss as the temperature is lowered, and in fact sometimes improve their performance. There seems to be a clear advantage for GIS procedures in sharply peaked distributions. Also they appear to have much more robustness against varying steepness in the underlying distribution. However, at warmer temperatures the MCMC methods are clearly superior.

It is important to note that greedy importance sampling is not equivalent to *adaptive* importance sampling. Sample blocks are completely independent in GIS, but sample points are not independent in AIS. Nevertheless, GIS can benefit from adapting the proposal distribution in the same way as standard IS. Clearly we cannot propose GIS methods as a replacement for MCMC approaches, and in fact believe that useful hybrid combinations are possible. Our goal in this research is to better understand a novel approach to estimation that appears to be worth investigating. Much work remains to be done in reducing computational overhead and investigating additional variance reduction techniques.

## Footnotes

[1]Unfortunately, for standard inference problems in graphical models it is usually not possible to evaluate $P(x)$ directly but rather just $\tilde{P}(x) = P(x) Z$ for some unknown constant $Z$. However it is still possible to apply the "indirect" importance sampling procedure shown in Figure 1 by assigning indirect weights $u(x) = \tilde{P}(x)/Q(x)$ and renormalizing. The drawback of the indirect procedure is that it is no longer unbiased at small sample sizes, but instead only becomes unbiased in the large sample limit [4]. To keep the presentation simple we will focus on the "direct" form of importance sampling described in Figure 1 and establish unbiasedness for that case—keeping in mind that every extended form of importance sampling we discuss below can be converted to an "indirect" form.

[2]There is no restriction on the blocks other than that they be finite—blocks can overlap and need not even contain their initiating point $x_i$—however their union has to cover the sample space $X$, and $Q$ cannot put zero probability on initiating points which leaves sample points uncovered.

[3]We merely chose the simplest heavy tailed convergent series available.

[4]This variance reduction scheme applies naturally to unbiased direct estimators. With indirect estimators, bias is typically more problematic than variance. Therefore, for indirect GIS we employ an alternative $\gamma$-weighting scheme that attempts to maximize total block weight.

[5]Interesting recent progress has been made on developing exact and approximate sampling methods for the special case of Ising models [9, 15, 13].

# References

[1] D. Ackley, G. Hinton, and T. Sejnowski. A learning algorithm for Boltzmann machines. *Cognitive Science*, 9:147–169, 1985.

[2] P. Dagum and M. Luby. Approximating probabilistic inference in Bayesian belief networks is NP-hard. *Artificial Intelligence*, 60:141–153, 1993.

[3] P. Dagum and M. Luby. An optimal approximation algorithm for Bayesian inference. *Artificial Intelligence*, 93:1–27, 1997.

[4] J. Geweke. Baysian inference in econometric models using Monte Carlo integration. *Econometrica*, 57:1317–1339, 1989.

[5] W. Gilks, S. Richardson, and D. Spiegelhalter. *Markov Chain Monte Carlo in Practice*. Chapman and Hall, 1996.

[6] M. Jordan, Z. Ghahramani, T. Jaakkola, and L. Saul. An introduction to variational methods for graphical models. In *Learning in Graphical Models*. Kluwer, 1998.

[7] D. MacKay. Intro to Monte Carlo methods. In *Learning in Graphical Models*. Kluwer, 1998.

[8] R. Neal. Probabilistic inference using Markov chain Monte Carlo methods. Tech report, 1993.

[9] J. Propp and D. Wilson. Exact sampling with coupled Markov chains and applications to statistical mechanics. *Random Structures and Algorithms*, 9:223–253, 1996.

[10] R. Rubinstein. *Simulation and the Monte Carlo Method*. Wiley, New York, 1981.

[11] D. Schuurmans. Greedy importance sampling. In *Proceedings NIPS-12*, 1999.

[12] D. Schuurmans and F. Southey. Monte Carlo inference via greedy importance sampling. In *Proceedings UAI*, 2000.

[13] R. Swendsen, J. Wang, and A. Ferrenberg. New Monte Carlo methods for improved efficiency of computer simulations in statistical mechanics. In *The Monte Carlo Method in Condensed Matter Physics*. Springer, 1992.

[14] M. Tanner. *Tools for Statistical Inference: Methods for Exploration of Posterior Distributions and Likelihood Functions*. Springer, New York, 1993.

[15] D. Wilson. Sampling configurations of an Ising system. In *Proceedings SODA*, 1999.
